# Eye movements and the maturation of cortical orientation selectivity

**Michele Rucci[1] and Antonino Casile[2]**
[1]Department of Cognitive and Neural Systems, Boston University, Boston, MA 02215.
[2]Scuola Superiore S. Anna, Pisa, Italy

## Abstract

Neural activity appears to be a crucial component for shaping the receptive fields of cortical simple cells into adjacent, oriented subregions alternately receiving ON- and OFF-center excitatory geniculate inputs. It is known that the orientation selective responses of V1 neurons are refined by visual experience. After eye opening, the spatiotemporal structure of neural activity in the early stages of the visual pathway depends both on the visual environment and on how the environment is scanned. We have used computational modeling to investigate how eye movements might affect the refinement of the orientation tuning of simple cells in the presence of a Hebbian scheme of synaptic plasticity. Levels of correlation between the activity of simulated cells were examined while natural scenes were scanned so as to model sequences of saccades and fixational eye movements, such as microsaccades, tremor and ocular drift. The specific patterns of activity required for a quantitatively accurate development of simple cell receptive fields with segregated ON and OFF subregions were observed during fixational eye movements, but not in the presence of saccades or with static presentation of natural visual input. These results suggest an important role for the eye movements occurring during visual fixation in the refinement of orientation selectivity.

## 1   Introduction

Cortical orientation selectivity, i.e. the preference to edges with specific orientations exhibited by most cells in the primary visual cortex of different mammal species, is one of the most investigated characteristics of neural responses. Although the essential elements of cortical orientation selectivity seem to develop before the exposure to patterned visual input, visual experience appears essential both for refining orientation selectivity, and maintaining the normal response properties of cortical neurons. The precise mechanisms by which visually-induced activity contribute to the maturation of neural responses are not known.

A number of experimental findings support the hypothesis that the development of orientation selective responses relies on Hebbian/covariance mechanisms of plasticity. According to this hypothesis, the stabilization of synchronously firing afferents onto common postsynaptic neurons may account for the segregation of neural inputs observed in the receptive fields of simple cells, where the adjacent oriented excitatory and inhibitory subregions re-

ceive selective input from geniculate ON- and OFF-center cells in the same retinotopic positions. Modeling studies [10, 9] have shown the feasibility of this proposal assuming suitable patterns of spontaneous activity in the LGN before eye opening.

After eye opening, the spatiotemporal structure of LGN activity depends not only on the characteristics of the visual input, but also on the movements performed by the animal while exploring its environment. It may be expected that changes in the visual input induced by these movements play an important role in shaping the responses of neurons in the visual system. In this paper we focus on how visual experience and eye movements might jointly influence the refinement of orientation selectivity under the assumption of a Hebbian mechanism of synaptic plasticity. As illustrated in Fig. 1, a necessary requirement of the Hebbian hypothesis is a consistency between the correlated activity of thalamic afferents and the organization of simple cell receptive fields. Synchronous activation is required among geniculate cells of the same type (ON- or OFF-center) with receptive fields located at distances smaller than the width of a simple cell subregion, and among cells of opposite polarity with receptive fields at distances comparable to the separation between adjacent subregions. We have analyzed the second order statistical structure of neural activity in a model of cat LGN when natural visual input was scanned so as to replicate the oculomotor behavior of the cat. Patterns of correlated activity were compared to the structure of simple cell receptive fields at different visual eccentricities.

## 2   The model

**Modeling the activity of LGN cells**

LGN cells were modeled as linear elements with quasi-separable spatial and temporal components as proposed by [3]. This model, derived using the reverse-correlation technique, has been shown to produce accurate estimates of the activity of different types of LGN cells. Changes in the instantaneous firing rates with respect to the level of spontaneous activity were generated by evaluating the spatiotemporal convolution of the input image $I$ with the receptive field kernel $K$

$$l_{xy}(t) = [K(x, y, t) \star I(x, y, t)]_\theta \tag{1}$$

where $\star$ is the symbol for convolution, $(x, y)$ and $t$ are the spatial and temporal variables, and the operator $[.]_\theta$ indicates rectification ($[x]_\theta = x - \theta$ if $x > \theta$, 0 otherwise). For each cell, the kernel $K$ consisted of two additive components, representing the center ($c$) and the periphery ($s$) of the receptive field respectively. Each of these two contributions was separable in its spatial ($F$) and temporal ($G$) elements:

$$K(x, y, t) = F_c(x, y)G_c(t) - F_s(x, y)G_s(t)$$

The spatial receptive fields of both center and surround were modeled as two-dimensional Gaussians, with a common space constant for both dimensions. Spatial parameters varied with eccentricity following neurophysiological measurements. As in [3], the temporal profile of the response was given by the difference of two gamma functions, with the temporal function for the periphery equal to that for the center and delayed by 3 ms.

**Modeling eye movements**

Modeled eye movements included saccades (both large-scale saccades and microsaccades), ocular drift, and tremor.

*Saccades*— Voluntary saccadic eye movements, the fast shifts of gaze among fixation points, were modeled by assuming a generalized exponential distribution of fixation times. The amplitude and direction of a saccade were randomly selected among all possible saccades that would keep the point of fixation on the image. Following data described in the

literature, the duration of each saccade was proportional to its amplitude. A modulation of geniculate activity was present in correspondence of each saccade [7]. Neural activity around the time of a saccade was multiplied by a gain function so that an initial suppression of activity with a peak of 10%, gradually reversed to a 20% facilitation with peak occurring 100 ms after the end of the saccade.

*Fixational eye movements—* Small eye movements included fixational saccades, ocular drift and tremor. Microsaccades were modeled in a similar way to voluntary saccades, with amplitude randomly selected from a uniform distribution between 1 and 10 minutes of arc. No modulation of LGN activity was present in the case of microsaccades.

Ocular drift and tremor were modeled together by approximating their power spectrum by means of a Poisson process filtered by a second order eye plant transfer function over the frequency range 0-40 Hz where the power declines as $1/f^2$. This term represents the irregular discharge rate of motor units for frequency less than 40 Hz. Parameters were adjusted so as to give a mean amplitude of $1.21^o$ and a mean velocity equal to $14.9^o$/s, which are the values measured in the cat [11].

## 3 Results

We simulated the activity of geniculate cells with receptive fields in different positions of the visual field, while receiving visual input in the presence of different types of eye movements. The relative level of correlation between units of the same and different types at positions $i$ and $j$ in the LGN was measured by means of the correlation difference, $C_{ij}^{D} = C_{ij}^{ONON} - C_{ij}^{ONOFF}$, where the two terms are the correlation coefficients evaluated between the two ON units at positions $i$ and $j$, and between the ON unit at position $i$ and the OFF unit at position $j$ respectively. $C_{ij}^{D}$ is positive when the activity of units of the same type covary more strongly than that of units of different types, and is negative when the opposite occurs. The average relative levels of correlation between units with receptive fields at different distances in the visual field were examined by means of the function $C^{D}(d) = < C_{ij}^{D} >_{|i-j|=d}$, which evaluates the average correlation difference $C_{ij}^{D}$ among all pairs of cells at positions $i$ and $j$ at distance $d$ from each other. For simplicity, in the following we refer to $C^{D}(d)$ as the correlation difference, implicitly assuming that a spatial averaging has taken place. The correlation difference is a useful tool for predicting the emerging patterns of connectivity in the presence of a Hebbian mechanism of synaptic plasticity. The average separation at which $C^{D}(d)$ changes sign is a key element in determining the spatial extent of the different subfields within the receptive fields of simple cells.

Fig. 1 (*b*) provides an example of application of the correlation difference function to quantify the correlated activity of LGN cells. In this example we have measured the level of correlation between pairs of cells with receptive fields at different separations when a spot of light was presented as input. An important element in the resulting level of correlation is the polarity of the two cells (*i.e.* whether they are ON- or OFF-center). As shown in Fig. 1 (*a*), since geniculate cells tend to be coactive when the ON and OFF subregions of their receptive fields overlap, the correlation between pairs of cells of the same type decreases when the separation between their receptive fields is increased, while pairs of cells of opposite types tend to become more correlated. As a consequence, the correlation difference function, $C^{D}(d)$, is positive at small separations, and negative at large ones.

Fig. 2 shows the measured correlated activity for LGN cells located around 17 deg. of visual eccentricity in the presence of two types of visual input: retinal spontaneous activity and natural visual stimulation. Spontaneous activity was simulated on the basis of Matronarde's data on the correlated firing of ganglion cells in the cat retina [8]. As illustrated by the graph, a close correspondence is present between the measured $C^{D}$ and the response profile of an average cortical simple cell at this eccentricity, indicating that a

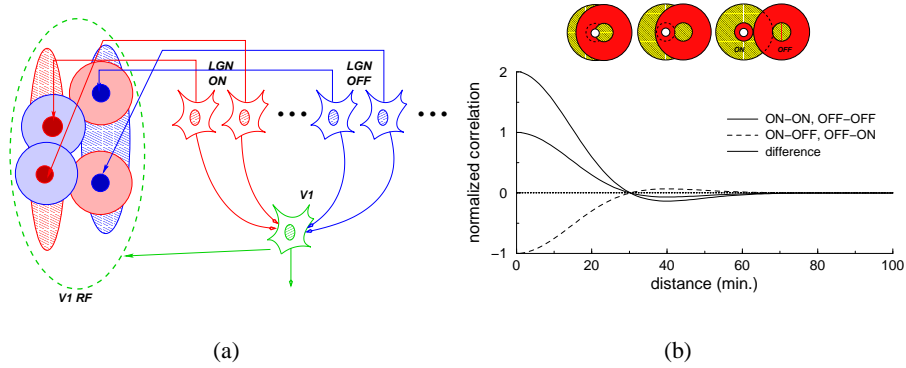

(a)                                          (b)

Figure 1: (a) Patterns of correlated activity required by a Hebbian mechanism of synaptic plasticity to produce a segregation of geniculate afferents. On average ON- and OFF-center LGN cells overlapping excitatory and inhibitory subregions in the receptive field of a simple cell must be simultaneously active. (b) Example of application of the correlation difference function, $C^{\mathrm{D}}(d)$. The icons on the top of each graph represent the positions of the receptive fields of the two cells at the corresponding separations along the $x$ axis. The bright dot marks the center of the spot of light. The three curves represent the correlation coefficients for pairs of units of the same type $C^s(d)$ (continuous thin line), units of opposite types $C^o(d)$ (dashed line), and the correlation difference function $C^{\mathrm{D}}(d) = C^s(d) - C^o(d)$ (bold line). Positive (negative) values of $C^{\mathrm{D}}(d)$ indicate that the activity of LGN cells of the same (opposite) type covary more closely than the activity of cells of opposite (same) types.

Hebbian mechanism of synaptic plasticity can well account for the structure of simple cell receptive fields before eye opening.

What happens in the presence of natural visual input? We evaluated the correlation difference function on a database of 30 images of natural scenes. The mean power spectrum of our database was best approximated by $S(k) = Aw^{-2.04}$, which is consistent with the results of several studies investigating the power spectrum of natural images. The mean correlation difference function measured when the input images were analyzed statically is marked by dark triangles in the left panel of Fig. 2. Due to the wide spatial correlations of natural visual input, the estimated correlation difference did not change sign within the receptive field of a typical simple cell. That is, LGN cells of the same type were found to covary more closely than cells of opposite types at all separations within the receptive field of a simple cell. This result is not consistent with the putative role of a direct Hebbian/covariance model in the refinement of orientation selectivity after eye opening.

A second series of simulations was dedicated to analyze the effects of eye movements on the structure of correlated activity. In these simulations the images of natural scenes were scanned so as to replicate cat oculomotor behavior. As shown in right panel of Fig. 2, significantly different patterns of correlated neural activity were found in the LGN in the presence of different types of eye movements. In the presence of large saccades, levels of correlations among the activity of geniculate cells were similar to the case of static presentation of natural visual input, and they did not match the structure of simple cell receptive fields. The dark triangles in Fig. 2 represent the correlation difference function evaluated over a window of observation of 100 ms in the presence of both large saccades and fixation eye movements. In contrast, when our analysis was restricted to the periods of visual fixation during which microscopic eye movements occurred, strong covariances were

measured between cells of the same type located nearby and between cells of opposite types at distances compatible with the separation between different subregions in the receptive fields of simple cells.

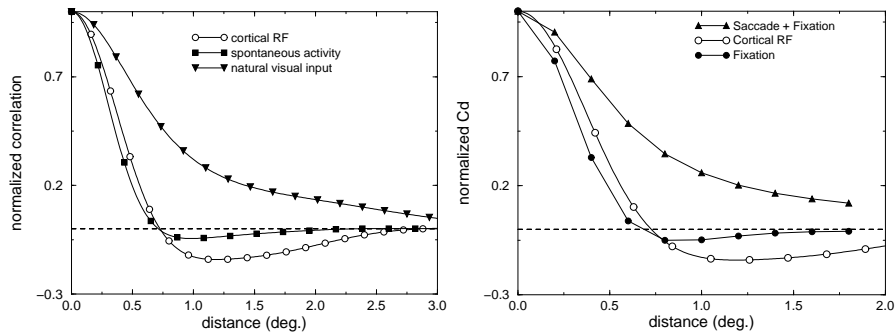

Figure 2: Analysis of the correlated activity of LGN units in different experimental conditions. In both graphs, the curve marked by white circles is the average receptive field of a simple cell, as measured by Jones and Palmer (1987) shown here for comparison. (*Left*) Static analysis: patterns of correlated activity in the presence of spontaneous activity and when natural visual input was analyzed statically. (*Right*) Effects of eye movements: correlation difference functions measured when natural images were scanned with sequence or saccades or fixational eye movements.

Fig. 3 shows the results of a similar analysis for LGN cells at different visual eccentricities. The white circles in the panels of Fig. 3 represent the width of the largest subfield in the receptive field of cortical simple cells as measured by [13]. The other curves on the left panel represent the widths of the central lobe of the correlation difference functions (the spatial separation over which cells of the same type possess correlated activity, measured as the double of the point in which the correlation difference function intersects the zero axis) in the cases of spontaneous activity and static presentation of natural visual input. As in Fig. 2, (1) a close correspondence was present between the experimental data and the subregion widths predicted by the correlation difference function in the case of spontaneous activity; and (2) a significant deviation between the two measurements was present in the case of static examination of natural visual input. The right panel in Fig. 3 shows the correlation difference functions obtained at different visual eccentricities in the presence of fixational eye movements. The minimum separation between receptive fields necessary for observing strong levels of covariance between cells with opposite polarity increased with eccentricity, as illustrated by the increase in the central lobe of the estimated correlation functions at the different visual eccentricities. As for the case of spontaneous activity, a close correspondence is now present between the spatiotemporal characteristics of LGN activity and the organization of simple cell receptive fields.

## 4 Discussion

In this paper we have used computer modeling to study the correlated activity of LGN cells when images of natural scenes were scanned so as to replicate cat eye movements. In the absence of eye movements, when a natural visual environment was observed statically, similar to the way it is examined by animals with their eyes paralyzed, we found that the simulated responses of geniculate cells of the same type at any separation smaller than the receptive field of a simple cell were strongly correlated. These spatial patterns of covarying geniculate activity did not match the structure of simple cell receptive fields. A similar result was obtained when natural scenes were scanned through saccades. Conversely, in

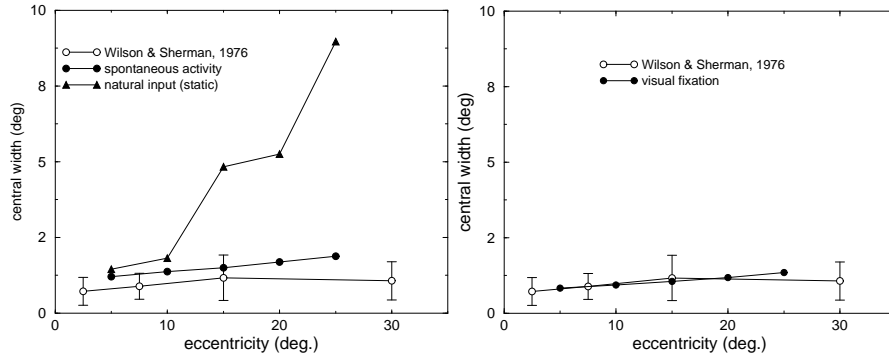

Figure 3: Analysis of the correlated activity of LGN units at different visual eccentricities. The width of the larger subfield in the receptive field of simple cells at different eccentricities as measured by Wilson and Sherman (1976) (white circles) is compared to the width of the central lobe of the correlation difference functions measured in different conditions (*Left*) Static analysis: results obtained in the presence of spontaneous activity and when natural visual input was analyzed statically. (*Right*) Case of fixational eye movements and natural visual input.

the case of micromovements, including both microsaccades and the combination of ocular drift and tremor, strong correlations were measured among cells of the same type located nearby and among cells of opposite types at distances compatible with the separation between different subregions in the receptive fields of simple cells. These results suggest a developmental role for the small eye movements that occur during visual fixation.

Although the role of visual experience in the development of orientation selectivity has been extensively investigated, relatively few studies have focused on whether eye movements contribute to the development of the responses of cortical cells. Yet, experiments in which kittens were raised with their eyes paralyzed have shown basic deficiencies in the development of visually-guided behavior [6], as well as impairments in ocular dominance plasticity [4, 12]. In addition, it has been shown that eye movements are necessary for the reestablishment of cortical orientation selectivity in dark-reared kittens exposed to visual experience within the critical period [2, 5]. This indicates that simultaneous experience of visual input and eye movements (and/or eye movement proprioception) may be necessary for the refinement of orientation selectivity [1]. Our finding that the patterns of LGN activity with static presentation of natural images did not match the spatial structure of the receptive fields of simple cells is in agreement with the hypothesis that exposure to pattern vision *per se* is not sufficient to account for a normal visual development.

A main assumption of this study is that the refinement and maintenance of orientation selectivity after eye opening is mediated by a Hebbian/covariance process of synaptic plasticity. The term Hebbian is used here with a generalized meaning to indicate the family of algorithms in which modifications of synaptic efficacies occur on the basis of the patterns of input covariances. While no previous theoretical study has investigated the influence of eye movements on the development of orientation selectivity, some models have shown that schemes of synaptic modifications based on the correlated activity of thalamic afferents can account well for the segregation of ON- and OFF-center inputs before eye opening in the presence of suitable patterns of spontaneous activity [10, 9]. By showing that, during fixation, the spatiotemporal structure of visually-driven geniculate activity is compatible with the structure of simple cell receptive fields, the results of the present study extend the plausibility of such schemes to the period after eye opening in which exposure to pattern

vision occurs.

Ocular movements are a common feature of the visual system of different species. It should not come as a surprise that a trace of their existence can be found even in some of the most basic properties of neurons in the early stages of the visual system, such as orientation selectivity. Further studies are needed to investigate whether similar traces can be found in other features of visual neural responses.

## References

[1] P. Buisseret. Influence of extraocular muscle proprioception on vision. *Physiol. Rev.*, 75(2):323–338, 1995.

[2] P. Buisseret, E. Gary-Bobo, and M. Imbert. Ocular motility and recovery of orientational properties of visual cortical neurons in dark-reared kittens. *Nature*, 272:816–817, 1978.

[3] D. Cai, G. C. DeAngelis, and R. D. Freeman. Spatiotemporal receptive field organization in the lateral geniculate nucleus of cats and kitten. *J. Neurophysiol.*, 78(2):1045–61, 1997.

[4] R. D. Freeman and A. B. Bonds. Cortical plasticity in monocularly deprived immobilized kittens depends on eye movement. *Science*, 206:1093–1095, 1979.

[5] E. Gary-Bobo, C. Milleret, and P. Buisseret. Role of eye movements in developmental process of orientation selectivity in the kitten visual cortex. *Vision Res.*, 26(4):557–567, 1986.

[6] A. Hein, F. Vital-Durand, W. Salinger, and R. Diamond. Eye movements initiate visual-motor development in the cat. *Science*, 204:1321–1322, 1979.

[7] D. Lee and J. G. Malpeli. Effect of saccades on the activity of neurons in the cat lateral geniculate nucleus. *J. Neurophysiol.*, 79:922–936, 1998.

[8] D. N. Mastronarde. Correlated firing of cat retinal ganglion cells. I spontaneously active inputs to X and Y cells. *J. Neurophysiol.*, 49(2):303–323, 1983.

[9] K. D. Miller. A model of the development of simple cell receptive fields and the ordered arrangement of orientation columns through activity-dependent competition between ON- and OFF- center inputs. *J. Neurosci.*, 14(1):409–441, 1994.

[10] M. Miyashita and S. Tanaka. A mathematical model for the self-organization of orientation columns in visual cortex. *Neuroreport*, 3:69–72, 1992.

[11] E. Olivier, A. Grantyn, M. Chat, and A. Berthoz. The control of slow orienting eye movements by tectoreticulospinal neurons in the cat: behavior, discharge patterns and underlying connections. *Exp. Brain Res.*, 93:435–449, 1993.

[12] W. Singer and J. Raushecker. Central-core control of developmental plasticity in the kitten visual cortex II. Electrical activation of mesencephalic and diencephalic projections. *Exp. Brain Res.*, 47:22–233, 1982.

[13] J. R. Wilson and S. M. Sherman. Receptive-field characteristics of neurons in the cat striate cortex: changes with visual field eccentricity. *J. Neurophysiol.*, 39(3):512–531, 1976.
